# Reinforcement Learning
# with Long Short-Term Memory

**Bram Bakker**
Dept. of Psychology, Leiden University / IDSIA
P.O. Box 9555; 2300 RB, Leiden; The Netherlands
*bbakker@fsw.leidenuniv.nl*

## Abstract

This paper presents reinforcement learning with a Long Short-Term Memory recurrent neural network: RL-LSTM. Model-free RL-LSTM using Advantage($\lambda$) learning and directed exploration can solve non-Markovian tasks with long-term dependencies between relevant events. This is demonstrated in a T-maze task, as well as in a difficult variation of the pole balancing task.

## 1 Introduction

Reinforcement learning (RL) is a way of learning how to behave based on delayed reward signals [12]. Among the more important challenges for RL are tasks where part of the state of the environment is *hidden* from the agent. Such tasks are called non-Markovian tasks or Partially Observable Markov Decision Processes. Many real world tasks have this problem of hidden state. For instance, in a navigation task different positions in the environment may look the same, but one and the same action may lead to different next states or rewards. Thus, hidden state makes RL more realistic. However, it also makes it more difficult, because now the agent not only needs to learn the mapping from environmental states to actions, for optimal performance it usually needs to determine which environmental state it is in as well.

**Long-term dependencies.** Most approaches to solving non-Markovian RL tasks have problems if there are long-term dependencies between relevant events. An example of a long-term dependency problem is a maze navigation task where the only way to distinguish between two T-junctions that look identical is to remember an observation or action a long time before either T-junction. Such a case presents obvious problems for fixed size history window approaches [6], which attempt to resolve the hidden state by making the chosen action depend not only on the current observation, but also on a fixed number of the most recent observations and actions. If the relevant piece of information to be remembered falls outside the history window, the agent cannot use it. McCallum's *variable* history window [8] has, in principle, the capacity to represent long-term dependencies. However, the system starts with zero history and increases the depth of the history window step by step. This makes learning long-term dependencies difficult, especially when there are no short-term dependencies to build on.

Other approaches to non-Markovian tasks are based on learning Finite State Automata [2], recurrent neural networks (RNNs) [10, 11, 6], or on learning to set

memory bits [9]. Unlike history window approaches, they do not have to represent (possibly long) entire histories, but can in principle extract and represent just the relevant information for an arbitrary amount of time. However, *learning* to do that has proven difficult. The difficulty lies in discovering the correlation between a piece of information and the moment at which this information becomes relevant at a later time, given the distracting observations and actions between them. This difficulty can be viewed as an instance of the general problem of learning long-term dependencies in timeseries data. This paper uses one particular solution to this problem that has worked well in *supervised* timeseries learning tasks: Long Short-Term Memory (LSTM) [5, 3]. In this paper an LSTM recurrent neural network is used in conjunction with model-free RL, in the same spirit as the model-free RNN approaches of [10, 6]. The next section describes LSTM. Section 3 presents LSTM's combination with reinforcement learning in a system called RL-LSTM. Section 4 contains simulation results on non-Markovian RL tasks with long-term dependencies. Section 5, finally, presents the general conclusions.

## 2   LSTM

LSTM is a recently proposed recurrent neural network architecture, originally designed for supervised timeseries learning [5, 3]. It is based on an analysis of the problems that conventional recurrent neural network learning algorithms, e.g. back-propagation through time (BPTT) and real-time recurrent learning (RTRL), have when learning timeseries with long-term dependencies. These problems boil down to the problem that errors propagated back in time tend to either vanish or blow up (see [5]).

**Memory cells.** LSTM's solution to this problem is to enforce *constant* error flow in a number of specialized units, called Constant Error Carrousels (CECs). This actually corresponds to these CECs having linear activation functions which do not decay over time. In order to prevent the CECs from filling up with useless information from the timeseries, access to them is regulated using other specialized, multiplicative units, called input gates. Like the CECs, the input gates receive input from the timeseries and the other units in the network, and they *learn* to open and close access to the CECs at appropriate moments. Access *from* the activations of the CECs to the output units (and possibly other units) of the network is regulated using multiplicative output gates. Similar to the input gates, the output gates learn when the time is right to send the information stored in the CECs to the output side of the network. A recent addition is forget gates [3], which learn to reset the activation of the CECs when the information stored in the CECs is no longer useful. The combination of a CEC with its associated input, output, and forget gate is called a memory cell. See figure 1b for a schematic of a memory cell. It is also possible for multiple CECs to be combined with only one input, output, and forget gate, in a so-called memory block.

**Activation updates.** More formally, the network's activations at each timestep $t$ are computed as follows. A standard hidden unit's activation $y^h$, output unit activation $y^k$, input gate activation $y^{in}$, output gate activation $y^{out}$, and forget gate activation $y^\varphi$ is computed in the following standard way:

$$y^i(t) = f_i(\sum_m w_{im} y^m(t-1)) \qquad (1)$$

where $w_{im}$ is the weight of the connection from unit $m$ to unit $i$. In this paper, $f_i$ is the standard logistic sigmoid function for all units except output units, for which it is the identity function. The CEC activation $s_{c_j^v}$, or the "state" of memory cell $v$

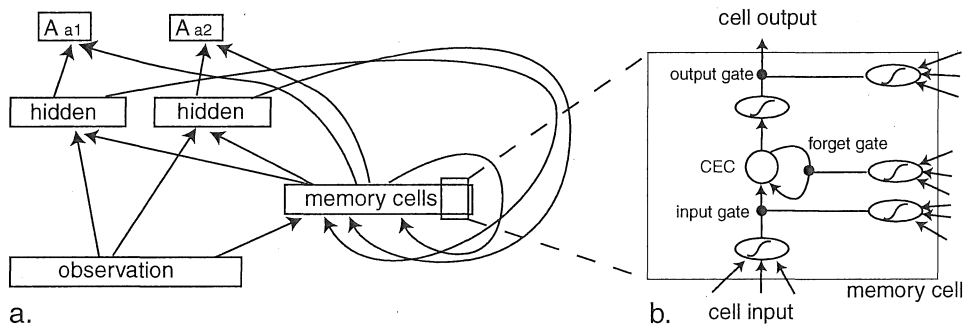

Figure 1: a. The general LSTM architecture used in this paper. Arrows indicate unidirectional, fully connected weights. The network's output units directly code for the Advantages of different actions. b. One memory cell.

in memory block $j$, is computed as follows:

$$s_{c_j^v}(t) = y^{\varphi_j}(t)s_{c_j^v}(t-1) + y^{in_j}(t)g(\sum_m w_{c_j^v m}y^m(t-1)) \qquad (2)$$

where $g$ is a logistic sigmoid function scaled to the range $[-2, 2]$, and $s_{c_j^v}(0) = 0$. The memory cell's output $y^{c_j^v}$ is calculated by

$$y^{c_j^v}(t) = y^{out_j}(t)h(s_{c_j^v}(t)) \qquad (3)$$

where $h$ is a logistic sigmoid function scaled to the range $[-1, 1]$.

**Learning.** At some or all timesteps of the timeseries, the output units of the network may make prediction errors. Errors are propagated just one step back in time through all units other than the CECs, including the gates. However, errors are backpropagated through the CECs for an indefinite amount of time, using an efficient variation of RTRL [5, 3]. Weight updates are done at every timestep, which fits in nicely with the philosophy of online RL. The learning algorithm is adapted slightly for RL, as explained in the next section.

## 3 RL-LSTM

RNNs, such as LSTM, can be applied to RL tasks in various ways. One way is to let the RNN learn a model of the environment, which learns to predict observations and rewards, and in this way learns to infer the environmental state at each point [6, 11]. LSTM's architecture would allow the predictions to depend on information from long ago. The model-based system could then learn the mapping from (inferred) environmental states to actions as in the Markovian case, using standard techniques such as Q-learning [6, 2], or by backpropagating through the frozen model to the controller [11]. An alternative, model-free approach, and the one used here, is to use the RNN to directly approximate the value function of a reinforcement learning algorithm [10, 6]. The state of the environment is approximated by the current observation, which is the input to the network, together with the recurrent activations in the network, which represent the agent's history. One possible advantage of such a model-free approach over a model-based approach is that the system may learn to only resolve hidden state insofar as that is *useful* for obtaining higher rewards, rather than waste time and resources in trying to predict features of the environment that are irrelevant for obtaining rewards [6, 8].

**Advantage learning.** In this paper, the RL-LSTM network approximates the value function of Advantage learning [4], which was designed as an improvement on

Q-learning for continuous-time RL. In continuous-time RL, values of adjacent states, and therefore optimal Q-values of different actions in a given state, typically differ by only small amounts, which can easily get lost in noise. Advantage learning remedies this problem by artificially decreasing the values of suboptimal actions in each state. Here Advantage learning is used for both continuous-time and discrete-time RL. Note that the same problem of small differences between values of adjacent states applies to any RL problem with long paths to rewards. And to demonstrate RL-LSTM's potential to bridge long time lags, we need to consider such RL problems. In general, Advantage learning may be more suitable for non-Markovian tasks than Q-learning, because it seems less sensitive to getting the value estimations exactly right.

The LSTM network's output units directly code for the Advantage values of different actions. Figure 1a shows the general network architecture used in this paper. As in Q-learning with a function approximator, the temporal difference error $E^{TD}(t)$, derived from the equivalent of the Bellman equation for Advantage learning [4], is taken as the function approximator's prediction error at timestep $t$:

$$E^{TD}(t) = V(s(t)) + \frac{r(t) + \gamma V(s(t+1)) - V(s(t))}{\kappa} - A(s(t), a(t)) \qquad (4)$$

where $A(s, a)$ is the Advantage value of action $a$ in state $s$, $r$ is the immediate reward, and $V(s) = \max_a A(s, a)$ is the value of the state $s$. $\gamma$ is a discount factor in the range $[0, 1]$, and $\kappa$ is a constant scaling the difference between values of optimal and suboptimal actions. Output units associated with other actions than the executed one do not receive error signals.

**Eligibility traces.** In this work, Advantage learning is extended with *eligibility traces*, which have often been found to improve learning in RL, especially in non-Markovian domains [7]. This yields Advantage($\lambda$) learning, and the necessary computations turn out virtually the same as in Q($\lambda$)-learning [1]. It requires the storage of one eligibility trace $e_{im}$ per weight $w_{im}$. A weight update corresponds to

$$w_{im}(t+1) = w_{im}(t) + \alpha E^{TD}(t) e_{im}(t) \text{ , where } e_{im}(t) = \gamma \lambda e_{im}(t-1) + \frac{\partial y^K(t)}{\partial w_{im}}. \quad (5)$$

$K$ indicates the output unit associated with the executed action, $\alpha$ is a learning rate parameter, and $\lambda$ is a parameter determining how fast the eligibility trace decays. $e_{im}(0) = 0$, and $e_{im}(t-1)$ is set to 0 if an exploratory action is taken.

**Exploration.** Non-Markovian RL requires extra attention to the issue of exploration [2, 8]. Undirected exploration attempts to try out actions in the same way in each environmental state. However, in non-Markovian tasks, the agent initially does not know which environmental state it is in. Part of the exploration must be aimed at discovering the environmental state structure. Furthermore, in many cases, the non-Markovian environment will provide unambiguous observations indicating the state in some parts, while providing ambiguous observations (hidden state) in other parts. In general, we want more exploration in the ambiguous parts.

This paper employs a directed exploration technique based on these ideas. A separate multilayer feedforward neural network, with the same input as the LSTM network (representing the current observation) and one output unit $y^v$, is trained concurrently with the LSTM network. It is trained, using standard backpropagation, to predict the absolute value of the current temporal difference error $E^{TD}(t)$ as defined by eq. 4, plus its own discounted prediction at the next timestep:

$$y_d^v(t) = |E^{TD}(t)| + \beta y^v(t+1) \qquad (6)$$

where $y_d^v(t)$ is the desired value for output $y^v(t)$, and $\beta$ is a discount parameter in the range $[0, 1]$. This amounts to attempting to identify which observations are

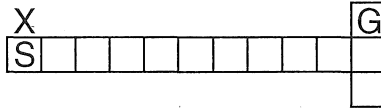

Figure 2: Long-term dependency T-maze with length of corridor $N = 10$. At the starting position S the agent's observation indicates where the goal position G is in this episode.

"problematic", in the sense that they are associated with large errors in the current value estimation (the first term), or precede situations with large such errors (the second term). $y^v(t)$ is linearly scaled and used as the temperature of a Boltzmann action selection rule [12]. The net result is much exploration when, for the current observation, differences between estimated Advantage values are small (the standard effect of Boltzmann exploration), or when there is much "uncertainty" about current Advantage values or Advantage values in the near future (the effect of this directed exploration scheme). This exploration technique has obvious similarities with the statistically more rigorous technique of Interval Estimation (see [12]), as well as with certain model-based approaches where exploration is greater when there is more uncertainty in the predictions of a model [11].

## 4   Test problems

**Long-term dependency T-maze.** The first test problem is a non-Markovian grid-based T-maze (see figure 2). It was designed to test RL-LSTM's capability to bridge long time lags, without confounding the results by making the control task difficult in other ways. The agent has four possible actions: move North, East, South, or West. The agent must learn to move from the starting position at the beginning of the corridor to the T-junction. There it must move either North or South to a changing goal position, which it cannot see. However, the location of the goal depends on a "road sign" the agent has seen at the starting position. If the agent takes the correct action at the T-junction, it receives a reward of 4. If it takes the wrong action, it receives a reward of $-.1$. In both cases, the episode ends and a new episode starts, with the new goal position set randomly either North or South. During the episode, the agent receives a reward of $-.1$ when it stands still. At the starting position, the observation is either 011 or 110, in the corridor the observation is 101, and at the T-junction the observation is 010. The length of the corridor $N$ was systematically varied from 5 to 70. In each condition, 10 runs were performed.

If the agent takes only optimal actions to the T-junction, it must remember the observation from the starting position for $N$ timesteps to determine the optimal action at the T-junction. Note that the agent is not aided by experiences in which there are shorter time lag dependencies. In fact, the opposite is true. Initially, it takes many more actions until even the T-junction is reached, and the experienced history is very variable from episode to episode. The agent must first learn to reliably move to the T-junction. Once this is accomplished, the agent will begin to experience more or less consistent and shortest possible histories of observations and actions, from which it can learn to extract the relevant piece of information. The directed exploration mechanism is crucial in this regard: it learns to set exploration low in the corridor and high at the T-junction.

The LSTM network had 3 input units, 12 standard hidden units, 3 memory cells, and $\alpha = .0002$. The following parameter values were used in all conditions: $\gamma = .98$, $\lambda = .8$, $\kappa = .1$. An empirical comparison was made with two alternative systems that have been used in non-Markovian tasks. The long-term dependency nature of the

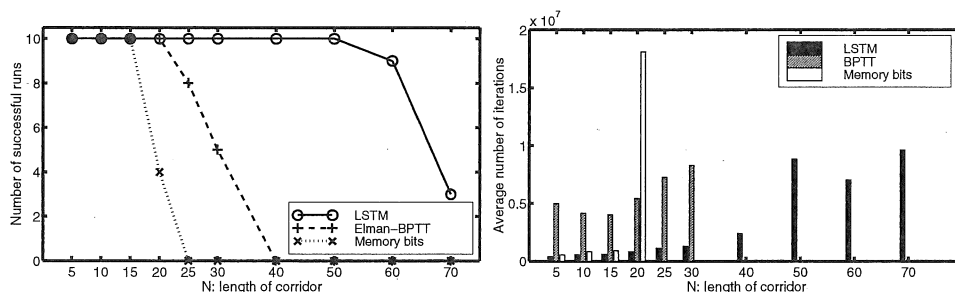

Figure 3: Results in noise-free T-maze task. Left: Number of successful runs (out of 10) as a function of $N$, length of the corridor. Right: Average number of timesteps until success as a function of $N$.

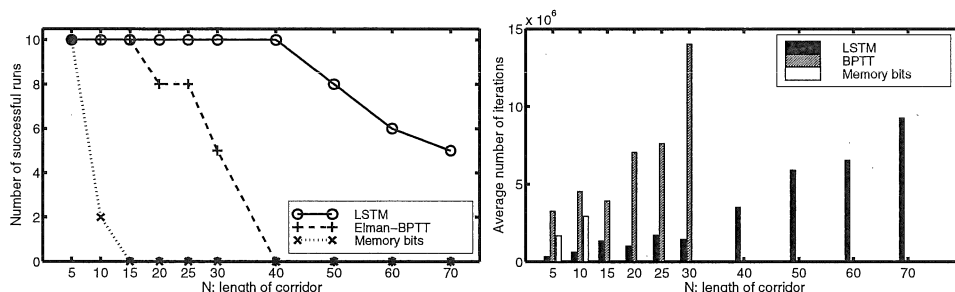

Figure 4: Results in noisy T-maze task. Left: Number of successful runs (out of 10) as a function of $N$, length of the corridor. Right: Average number of timesteps until success as a function of $N$.

task virtually rules out history window approaches. Instead, two alternative systems were used that, like LSTM, are capable in principle of representing information for arbitrary long time lags. In the first alternative, the LSTM network was replaced by an Elman-style Simple Recurrent Network, trained using BPTT [6]. Note that the unfolding of the RNN necessary for BPTT means that this is no longer truly online RL. The Elman network had 16 hidden units and 16 context units, and $\alpha = .001$. The second alternative is a table-based system extended with memory bits that are part of the observation, and that the controller can switch on and off [9]. Because the task requires the agent to remember just one bit of information, this system had one memory bit, and $\alpha = .01$. In order to determine the specific contribution of LSTM to performance, in both alternatives all elements of the overall system except LSTM (i.e. Advantage($\lambda$) learning, directed exploration) were left unchanged.

A run was considered a *success* if the agent learned to take the correct action at the T-junction in over 80% of cases, using its stochastic action selection mechanism. In practice, this corresponds to 100% correct action choices at the T-junction using greedy action selection, as well as optimal or near-optimal action choices leading to the T-junction. Figure 3 shows the number of successful runs (out of 10) as a function of the length of the corridor $N$, for each of the three methods. It also shows the average number of timesteps needed to reach success. It is apparent that RL-LSTM is able to deal with much longer time lags than the two alternatives. RL-LSTM has perfect performance up to $N = 50$, after which performance gradually decreases. In those cases where the alternatives also reach success, RL-LSTM also learns faster. The reason why the memory bits system performs worst is probably that, in contrast with the other two, it does not explicitly compute the gradient of performance with respect to past events. This should make credit assignment

less directed and therefore less effective. The Elman-BPTT system does compute such a gradient, but in contrast to LSTM, the gradient information tends to vanish quickly with longer time lags (as explained in section 2).

**T-maze with noise.** It is one thing to learn long-term dependencies in a noise-free task, it is quite another thing to do so in the presence of severe noise. To investigate this, a very noisy variation of the T-maze task described above was designed. Now the observation in the corridor is a0b, where a and b are independent, uniformly distributed random values in the range [0, 1], generate online. All other aspects of the task remain the same as above. Both the LSTM and the Elman-BPTT system were also left unchanged. To allow for a fair comparison, the table-based memory bit system's observation was computed using Michie and Chambers's BOXES state aggregation mechanism (see [12]), partitioning each input dimension into three equal regions.

Figure 4 shows the results. The memory bit system suffers most from the noise. This is not very surprising because a table-based system, even if augmented with BOXES state aggregation, does not give very sophisticated generalization. The two RNN approaches are hardly affected by the severe noise in the observations. Most importantly, RL-LSTM again significantly outperforms the others, both in terms of the maximum time lag it can deal with, and in terms of the number of timesteps needed to learn the task.

**Multi-mode pole balancing.** The third test problem is less artificial than the T-mazes and has more complicated dynamics. It consists of a difficult variation of the classical pole balancing task. In the pole balancing task, an agent must balance an inherently unstable pole, hinged to the top of a wheeled cart that travels along a track, by applying left and right forces to the cart. Even in the Markovian version, the task requires fairly precise control to solve it.

The version used in this experiment is made more difficult by two sources of hidden state. First, as in [6], the agent cannot observe the state information corresponding to the cart velocity and pole angular velocity. It has to learn to approximate this (continuous) information using its recurrent connections in order to solve the task. Second, the agent must learn to operate in two different modes. In mode A, action 1 is left push and action 2 is right push. In mode B, this is reversed: action 1 is *right* push and action 2 is *left* push. Modes are randomly set at the beginning of each episode. The information which mode the agent is operating in is provided to the agent only for the first second of the episode. After that, the corresponding input unit is set to zero and the agent must *remember* which mode it is in. Obviously, failing to remember the mode leads to very poor performance. The only reward signal is $-1$ if the pole falls past $\pm 12°$ or if the cart hits either end of the track. Note that the agent must learn to remember the (discrete) mode information for an infinite amount of time if it is to learn to balance the pole indefinitely. This rules out history window approaches altogether. However, in contrast with the T-mazes, the system now has the benefit of starting with relatively short time lags.

The LSTM network had 2 output units, 14 standard hidden units, and 6 memory cells. It has 3 input units: one each for cart position and pole angle; and one for the mode of operation, set to zero after one second of simulated time (50 timesteps). $\gamma = .95$, $\lambda = .6$, $\kappa = .2$, $\alpha = .002$. In this problem, directed exploration was not necessary, because in contrast to the T-mazes, imperfect policies lead to many different experiences with reward signals, and there is hidden state everywhere in the environment. For a continuous problem like this, a table-based memory bit system is not suited very well, so a comparison was only made with the Elman-BPTT system, which had 16 hidden and context units and $\alpha = .002$.

The Elman-BPTT system never reached satisfactory solutions in 10 runs. It only learned to balance the pole for the first 50 timesteps, when the mode information is available, thus failing to learn the long-term dependency. However, RL-LSTM learned optimal performance in 2 out of 10 runs (after an average of 6,250,000 timesteps of learning). After learning, these two agents were able to balance the pole indefinitely in both modes of operation. In the other 8 runs, the agents still learned to balance the pole in both modes for hundreds or even thousands of timesteps (after an average of 8,095,000 timesteps of learning), thus showing that the mode information was remembered for long time lags. In most cases, such an agent learns optimal performance for one mode, while achieving good but suboptimal performance in the other.

## 5   Conclusions

The results presented in this paper suggest that reinforcement learning with Long Short-Term Memory (RL-LSTM) is a promising approach to solving non-Markovian RL tasks with long-term dependencies. This was demonstrated in a T-maze task with minimal time lag dependencies of up to 70 timesteps, as well as in a non-Markovian version of pole balancing where optimal performance requires remembering information indefinitely. RL-LSTM's main power is derived from LSTM's property of constant error flow, but for good performance in RL tasks, the combination with Advantage($\lambda$) learning and directed exploration was crucial.

### Acknowledgments

The author wishes to thank Edwin de Jong, Michiel de Jong, Gwendid van der Voort van der Kleij, Patrick Hudson, Felix Gers, and Jürgen Schmidhuber for valuable comments.

### References

[1] B. Bakker. Reinforcement learning with LSTM in non-Markovian tasks with long-term dependencies. Technical report, Dept. of Psychology, Leiden University, 2001.

[2] L. Chrisman. Reinforcement learning with perceptual aliasing: The perceptual distinctions approach. In *Proc. of the 10th National Conf. on AI*. AAAI Press, 1992.

[3] F. Gers, J. Schmidhuber, and F. Cummins. Learning to forget: Continual prediction with LSTM. *Neural Computation*, 12 (10):2451–2471, 2000.

[4] M. E. Harmon and L. C. Baird. Multi-player residual advantage learning with general function approximation. Technical report, Wright-Patterson Air Force Base, 1996.

[5] S. Hochreiter and J. Schmidhuber. Long short-term memory. *Neural Computation*, 9 (8):1735–1780, 1997.

[6] L.-J. Lin and T. Mitchell. Reinforcement learning with hidden states. In *Proc. of the 2nd Int. Conf. on Simulation of Adaptive Behavior*. MIT Press, 1993.

[7] J. Loch and S. Singh. Using eligibility traces to find the best memoryless policy in Partially Observable Markov Decision Processes. In *Proc. of ICML'98*, 1998.

[8] R. A. McCallum. Learning to use selective attention and short-term memory in sequential tasks. In *Proc. 4th Int. Conf. on Simulation of Adaptive Behavior*, 1996.

[9] L. Peshkin, N. Meuleau, and L. P. Kaelbling. Learning policies with external memory. In *Proc. of the 16th Int. Conf. on Machine Learning*, 1999.

[10] J. Schmidhuber. Networks adjusting networks. In *Proc. of Distributed Adaptive Neural Information Processing*, St. Augustin, 1990.

[11] J. Schmidhuber. Curious model-building control systems. In *Proc. of IJCNN'91*, volume 2, pages 1458–1463, Singapore, 1991.

[12] R. S. Sutton and A. G. Barto. *Reinforcement learning: An introduction*. MIT Press, Cambridge, MA, 1998.
